# Unsupervised Learning of a Probabilistic Grammar for Object Detection and Parsing

**Long (Leo) Zhu**
Department of Statistics
University of California at Los Angeles
Los Angeles, CA 90095
lzhu@stat.ucla.edu

**Yuanhao Chen**
Department of Automation
University of Science and Technology of China
Hefei, Anhui 230026 P.R.China
yhchen4@ustc.edu

**Alan Yuille**
Department of Statistics
University of California at Los Angeles
Los Angeles, CA 90095
yuille@stat.ucla.edu

## Abstract

We describe an unsupervised method for learning a probabilistic grammar of an object from a set of training examples. Our approach is invariant to the scale and rotation of the objects. We illustrate our approach using thirteen objects from the Caltech 101 database. In addition, we learn the model of a hybrid object class where we do not know the specific object or its position, scale or pose. This is illustrated by learning a hybrid class consisting of faces, motorbikes, and airplanes. The individual objects can be recovered as different aspects of the grammar for the object class. In all cases, we validate our results by learning the probability grammars from training datasets and evaluating them on the test datasets. We compare our method to alternative approaches. The advantages of our approach is the speed of inference (under one second), the parsing of the object, and increased accuracy of performance. Moreover, our approach is very general and can be applied to a large range of objects and structures.

## 1 Introduction

Remarkable progress in mathematics and computer science of probability is leading to a revolution in the scope of probabilistic models. In particular, there are exciting new probability models [1, 3, 4, 5, 6, 11] defined on structured relational systems such as graphs or grammars. These new formulations subsume more traditional models, such as Markov Random Fields (MRF's) [2], and have growing applications to natural languages, machine learning, and computer vision.

Although these models have enormous representational power, there are many practical drawbacks which must be overcome before using them. In particular, we need efficient algorithms to learn the models from training data and to perform inference on new examples. This problem is particularly difficult when the structure of the representation is unknown and needs to be induced from the data.

In this paper we develop an algorithm called "structure induction" (or "structure pursuit") which we use to learn the probability model in an unsupervised manner from a set of training data. This algorithm proceeds by building an AND-OR graph [5] in an iterative way. The form of the resulting graph structure ensures that inference can be performed rapidly for new data.

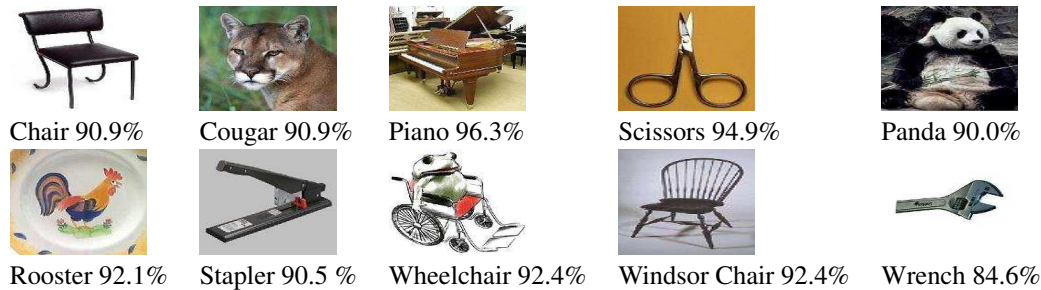

Chair 90.9%    Cougar 90.9%    Piano 96.3%    Scissors 94.9%    Panda 90.0%

Rooster 92.1%    Stapler 90.5 %    Wheelchair 92.4%    Windsor Chair 92.4%    Wrench 84.6%

Figure 1: We have learnt probability grammars for these ten objects in the Caltech 101 database, obtaining scores over 90 % for most objects. A score of 90.00 %, means that we have a detection rate of 90 % and a false positive rate of 10 % (10 % = (100 - 90) %). The number of data examples are 62, 69, 90, 39, 38, 49, 45, 59, 56, 39 ordered left-to-right and top-to-bottom.

Our application is to the detection, recognition, and parsing of objects in images. The training data consists of a set of images where the target object is present but at an unknown location. This topic has been much studied [16] (see technical report – Zhu, Chen and Yuille 2006 – for additional references).

Our approach has the following four properties. Firstly, a wide range of applicability which we demonstrate by learning models for 13 object categories from the Caltech-101 [16], Figure (1,5). Secondly, the approach is invariant to rotation and a large range of scale of the objects. Thirdly, the approach is able to deal with object classes, which we illustrate by learning a hybrid class consisting of faces, motorbikes and airplane. Fourthly, the inference is performed rapidly in under a second.

## 2 Background

### 2.1 Representation, Inference and Learning

Structured models define a probability distribution on structured relational systems such as graphs or grammars. This includes many standard models of probability distributions defined on graphs – for example, graphs with fixed structure, such as MRF's [2] or Conditional Random Fields [3], or Probabilistic Context Free Grammars (PCFG's) [4] where the structure is variable. Attempts have been made to unify these approaches under a common formulation. For example, Case-Factor Diagrams [1] have recently been proposed as a framework which subsumes both MRF's and PCFG's. In this paper, we will be concerned with models that combine probabilistic grammars with MRF's. The grammars are based on AND-OR graphs [1, 5, 6], which relate to mixtures of trees [7]. This merging of MRF's with probabilistic grammars results in structured models which have great representational power.

There has been considerable interest in inference algorithms for these structured models, for example McAllester *et al* [1] describe how dynamic programming algorithms (e.g. Viterbi and inside-outside) can be used to rapidly compute properties of interest. Our paper is concerned with the task of unsupervised learning of structured models for applications to detecting, recognizing, and representing visual objects. In this paper, we restrict ourselves to a special case of Probabilistic Grammars with OR nodes, and MRF's. This is simpler than the full cases studied by McAllester but is more complex than the MRF models standardly used for this problem.

For MRF models, the number of graph nodes is fixed and structure induction consists of determining the connections between the nodes and the forms of the corresponding potentials. For these graphs, an effective strategy is feature induction [8] which is also known as feature pursuit [9]. A similar strategy is also used to learn CRF's [10]. In both cases, the learning is fully supervised. For Bayesian network, there is work on learning the structure using the EM algorithm [12].

Learning the structure of grammars in an unsupervised way is more difficult. Klein and Manning [4] have developed unsupervised learning of PCFG's for parsing natural language, but here the structure of grammar is specified. Zettlemoyer and Collins [11] perform similar work based on lexical learning with lambda-calculus language.

In short, to our knowledge, there is no unsupervised learning algorithm for structure induction for a Probabilistic Grammar-MRF model. Moreover, our vision application requires the ability to learn the model of the target object in the presence of unknown background structure. Methods exist in the computer vision literature for achieving this for an MRF model [16], but not for Probabilistic Grammars.

## 2.2 Our Model: High-Level Description

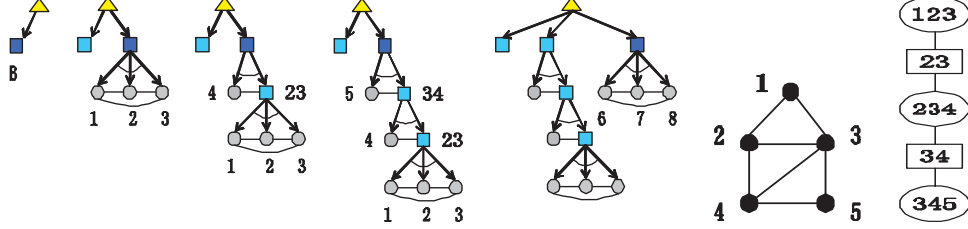

Figure 2: Graphical Models.

In this paper, we consider a combination of PCFG and MRF. The leaf nodes of the graph will be image features that are described by MRF's. Instead of using the full PCFG, we restrict the grammar to containing one OR-node.

Our model contains a restricted set of grammatical rules, see figure (2). The top, triangular node, is an OR node. It can have an arbitrary number of child nodes. The simplest type of child node is a histogram model (far left panel of figure (2)). We can obtain more complex models by adding MRF models in the form of triples, see figure (2) left to right. Combination of triples can be expressed in a junction tree representation, see the sixth and seventh panels of figure (2). This representation enable rapid inference. The computation complexity of inference is bounded by the width and height of the subtrees.

In more abstract terms, we define a set of rules $R(x, y)$ for allowable parses of input $x$ to a parse tree $y$. These rules have potentials $\phi(x, r, t)$ for a production rule $r \in R(x, y)$ and $\psi(x, w_M, t)$ for the MRF models (see details in the technical report), where $t$ are nuisance parameters (e.g. geometric transformations and missing data) and $w = (w_G, w_M)$ are model parameters. The $w_G$ are the grammar parameters and the $w_M$ are the MRF parameters. We define a set $W$ of model parameters that are allowed to be non-zero ($w = 0$ if $w \notin W$). The structure of the model is determined by the set $W$.

The model is defined by:

$$P(x, y, w, t) = P(t)P(w)P(y)P(x|y, w, t),\tag{1}$$

where

$$P(x|y, w, t) = \frac{1}{Z} e^{\sum_{r \in R(x,y)} w_G \cdot \phi(x,r,t) + \sum_{MRF} \Psi_{MRF}(x,t,w_M)},\tag{2}$$

where $_{MRF}$ denotes the cliques of the MRF. $Z$ is the normalization constant.

We now face three tasks: (I) structure learning, (II) parameter learning to estimate $w$, and (III) inference to estimate $y$.

**Inference** requires estimating the parse tree $y$ from input $x$. The model parameters $w$ are fixed. The nuisance parameters are integrated out. This requires solving $y^* = \arg\max \sum_t P(y, t|x, w)$ by the EM algorithm using dynamic programming to estimate $y^*$ efficiently. During the E step, we approximate the sum over $t$ by a saddle point approximation.

**Parameter learning** we specify a set $W$ of parameters $w$ which we estimate by MAP (the other $w$'s are constrained to be zero). Hence we estimate $w^* = \arg\max_{w \in W} \sum_{y,t} P(w, t, y|x)$. This is performed by an EM algorithm, where the summation over $y$ can be performed by dynamic programming, the summation over $t$ is again performed by a saddle point. The $w$ can be calculated by sufficient statistics.

**Structure Learning** corresponds to increasing the set of parameters $w$ that can be non-zero. For each structure we define a *score* given by its fit to the data. Formally we extend $W$ to $W'$ where $W \subset W'$. (The allowed extensions are defined in the next section). We now compute $P(x|w \in W) = \sum_{w \in W, t, y} P(x, y, t|w)$ and $P(x|w \in W') = \sum_{w \in W', t, y} P(x, y, t|w)$. This requires EM, dynamic programming, and a saddle point approximation. We refer to the model fits, $P(x|w \in W)$ and $P(x|w \in W')$, as the *scores* for structure $W$ and $W'$ respectively.

## 3  Brief Details of Our Model

We now give a brief description of our model. A detailed description is given in our technical report (Zhu, Chen, and Yuille 2006).

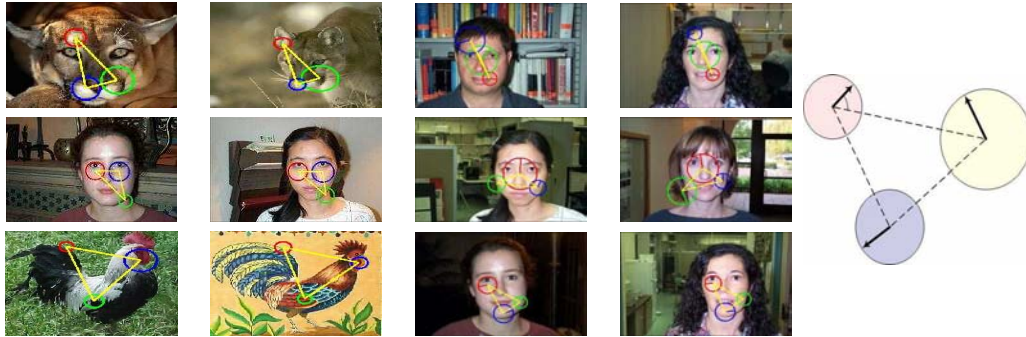

Figure 3: Triplets without Orientation (left two panels). Triplets with Orientation (right two panels).

### 3.1  The setup of the Model

We represent the images by features $\{x_i : i = 1, .., N(\tau)\}$, where $N(\tau)$ is the number of features in image $\tau$. Each feature is represented by a pair $x_i = (z_i, A_i)$, where $z_i$ is the location of the feature in the image and $A_i$ is an appearance vector. The image features are detected by the Kadir-Brady operator [13], and their appearance is calculated by the SIFT operator [14]. These operators ensure that the features are invariant to scale, rotation, and some appearance variations.

The default *background model* for the image is to define a histogram model over the positions and appearance of the image features, see first panel of figure (2).

Next we use triples of image features as the basic building blocks to construct a model. Our model will be constructed by adding new triplets to the existing model, as shown in the first few panels of figure (2). Each triplet will be represented by a *triplet model* which is given by Gaussian distributions on spatial position and on appearance $P(\vec{x}|\vec{M} = \vec{1}, \mathbf{T}) = G(\vec{z}|\mathbf{T}(\vec{\mu}_G, \mathbf{\Sigma_G})G(\vec{A}|\vec{\mu}_A, \mathbf{\Sigma_A})$, where $\mu_G, \mu_A, \Sigma_G, \Sigma_A$ are the means and covariances of the positions and appearances. The $\{M_i\}$ are missing data index variables [15], and $\mathbf{T}$ denotes transformations due to rotation and scaling.

The major advantage of using triplets is that they have geometrical properties which are independent of the scale and rotation of the triplet. These properties include the angles between the vertices, see figure (3). Thus we can decompose the representation of the triplet into two types of properties: (i) those which are independent of scale and rotation, (ii) those that depend explicitly on scale and rotation. By using the invariant properties, we can perform rapid search over triplets when position, scale, and rotation are unknown.

In addition, two triplets can be easily combined by a common edge to form a more complex model – see sixth panel of figure (2). This representation is suitable for the junction tree algorithm [2], which enables rapid inference.

For structure learning, we face the task of how to expand the set $W$ of non-zero parameters to a new set $W'$. The problem is that there are many ways to expand the set, and it is computationally impossible to evaluate all of them. Our strategy is to use a clustering method, see below, to make proposals for expanding the structure. These proposals are then evaluated by model selection.

Our clustering method exploits the invariance properties of triplets. We perform clustering on both the appearance and on the geometrical invariants of the triplets. This gives rise to a *triplet vocabulary* consisting of triplets that frequently occur in the dataset. These are used to make proposals for which triplets to include in the model, and hence for how to expand the set $W$ of non-zero parameters.

---

**Input:** Training Image $\tau = 1, .., M$ and the triplet vocabulary $\{T_a : a \in \Omega\}$. Initialize $G$ to be the root node with the background model, and let $G^* = G$.

**Algorithm for Structure Induction:**

- **STEP 1**:
    - OR-NODE EXTENSION
      For $T \in \{T_a : a \in \Omega\}$
        * $G' = G \bigcup T$ (ORing)
        * Update parameters of $G'$ by EM algorithm
        * If $Score(G') > Score(G^*)$ Then $G^* = G'$
    - AND-NODE EXTENSION
      For Image $\tau = 1, .., M$
        * P = the highest probability parse for Image $\tau$ by $G$
        * For each Triple $T$ in Image $\tau$
          if $T \bigcap P \neq \emptyset$
          · $G' = G \bigcup T$ (ANDing)
          · Update parameters of $G'$ by EM algorithm
          · If $Score(G') > Score(G^*)$ Then $G^* = G'$
- **STEP 2:** $G = G^*$. Go to STEP 1 until $Score(G) - Score(G^*) < Threshold$

**Output:** $G$

---

Figure 4: Structure Induction Algorithm

## 3.2  Structure Induction: Learning the Probabilistic Grammar MRF

We now have the necessary background to describe our structure induction algorithm. The full procedure is described in the pseudo code in figure (4). Figure (2) shows an example of the structure being induced sequentially.

Initially we assume that all the data is generated by the background model. In the terminology of section (2.2), this is equivalent to setting all of the model parameters $w$ to be zero (except those for the background model). We can estimate the parameters of this model and score the model as described in section (2.2).

Next we seek to expand the structure of this model. To do this, we use the triplet vocabularies to make proposals. Since the current model is the background model, the only structure change allowed is to add a triplet model as one child of the root node (i.e. to create the background plus triple model described in the previous section, see figure (2)). We consider all members of the triplet vocabulary as candidates, using their cluster means and covariances as prior probabilities on their geometry and attribute properties. Then, for all these triples we construct the background plus triplet model, estimate their parameters and score them. We accept the one with highest score as the new structure.

As the graph structure grows, we now have more ways to expand the graph. We can add a new triplet as a child of the root node. This proceeds as in the previous paragraph. Or we can take two members of an existing triplet, and use them to construct a new triplet. In this case, we first parse the data using the current model. Then we use the triplet vocabulary to propose possible triplets, which partially overlap with the current model (and give them prior probabilities on their parameters as before). Then, for all possible extensions, we use the methods in section (2.2) to score the models. We select the one with highest score as the new graph model. If the score increase is not sufficient, we cease building the graph model. See the structured models in figure (5).

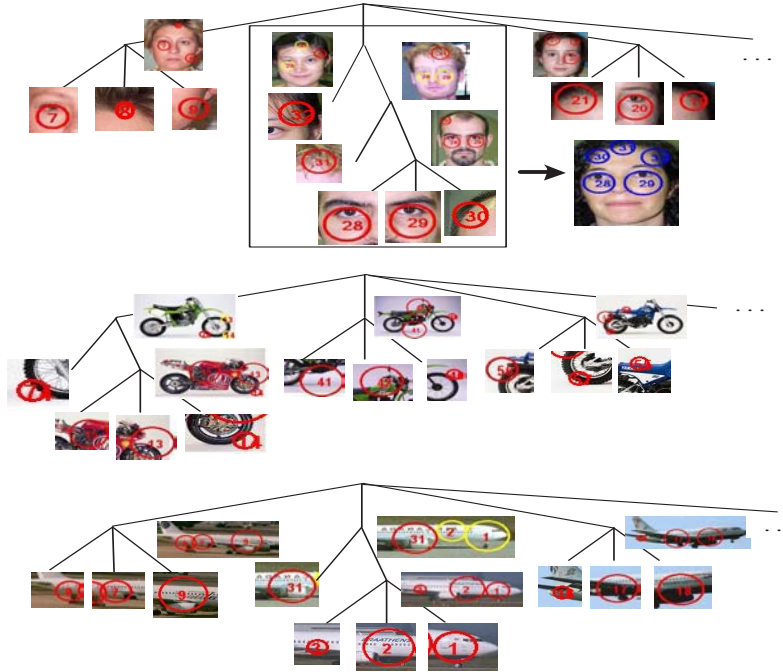

Figure 5: Individual Models learnt for Faces, Motorbikes and Airplanes.

Table 1: Performance Comparisons

| Dataset | Size | Single Model | Hybrid Model | Constellation[16] |
|---|---|---|---|---|
| Faces | 435 | 98.0 | 84.0 | 96.4 |
| Motorbikes | 800 | 92.6 | 82.7 | 92.5 |
| Airplanes | 800 | 90.9 | 87.3 | 90.2 |
| Faces(Rotated) | 435 | 94.8 | – | – |
| Faces(Rotated+Scaled) | 435 | 92.3 | – | – |

## 4    Experimental Results

### 4.1    Learning Individual Objects Models

In this section, we demonstrate the performance of our models for thirteen objects chosen from the Caltech-101 dataset. Each dataset was randomly split into two sets with equal size(one for training and the other for testing).

K-means clustering (typically, K is set to 150) was used to learn the triplet vocabularies (see Zhu, Chen, Yuille 2006 for details). Each row in figure 3 corresponds to some triples in the same group. In this experiment, we did not use orientation information from the feature detector.

We illustrate our results in figure (1) and Table (1). A score of 90 % means that we get a true positive rate of 90 % and a false positive rate of 10 %. For comparison, we show the performance of the Constellation Model [16]. (Further comparisons to alternative methods are reported in the technical report).

The models for individual objects classes, learnt from the proposed algorithm, are illustrated in figure (5). Observe that the generative models have different tree-width and depth. Each subtree of the root node defines an Markov Random Field to describe one configuration of the object. The computational cost of the inference, using dynamic programming, is proportional to the height of the subtree and exponential to the maximum width(only three in our case). The detection time is

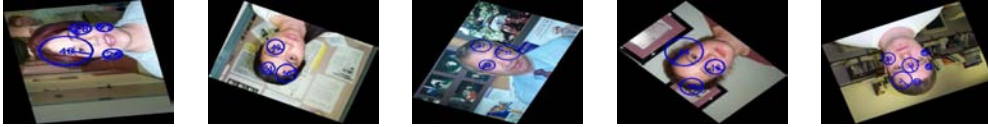

Figure 6: Parsed Results: Invariant to Rotation and Scale.

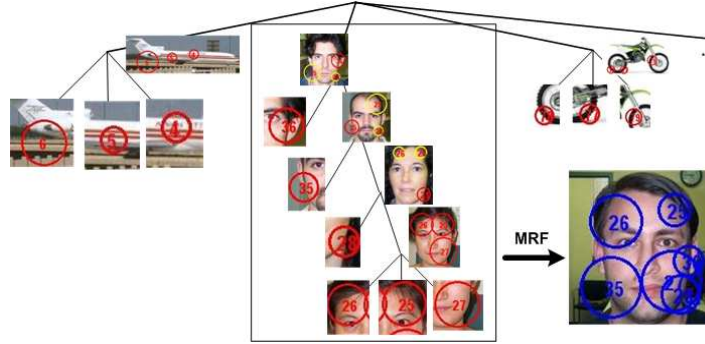

Figure 7: Hybrid Model learnt for Faces, Motorbikes and Airplanes.

less than one second (including the processing of features and inference) for the image with the size of 320*240. The training time is around two hours for 250 training images.

## 4.2 Invariance to Rotation and Scale

This section shows that we can learn and detect objects even when the rotation (in the image) and the scale are unknown (within a range). In this experiment, orientation information, output from feature detector, is used to model the geometry distributions of the triplets. The relative angle between the orientation of each feature and the orientation of the edge of tri-angle is calculated to make the model invariant to rotation. See Figure (3).

We implemented the comparison experiment on face dataset. A face model is learnt from the training images with normalized scale and orientation. We tested this model on the testing data with 360-degree in-plane rotation and another testing data with rotation and scaling together. The scaling range is from 60% of the original size to 150%(i.e. $180 * 120 - 450 * 300$). Table (1) shows the comparison results. The parsing results (rotation+scale) are illustrated in Figure (6).

## 4.3 Learning Classes of Models

In this section, we show that we can learn a model for an object class. We use a hybrid class which consists of faces, airplanes, and motorbikes. In other words, we know that one object is present in each image but we do not know which. In the training stage, we randomly select images from the datasets of faces, airplanes, and motorbikes. Similarly, we test the hybrid model on examples selected randomly from these three datasets.

The learnt hybrid model is illustrated in Figure (7). It breaks down nicely into or's of the models for each object. Table (1) shows the performance for the hybrid model. This demonstrates that the proposed method can learn a model for the class with extremely large variation. The parsed results are shown in Figure (8).

## 5 Discussion

This paper showed that it is possible to perform unsupervised learning to determine a probabilistic grammar combined with a Markov Random Fields. Our approach is based on structure pursuit where the object model is built up in an iterative manner (similar to feature pursuit used for MRF's and CRF's). The building blocks of our model are triplets of features, whose invariance properties can be exploited for rapid computation.

Our application is to the detection and parsing of objects. We demonstrated: (a) that we can learn probabilistic models for a variety of different objects, (b) that our approach is invariant to scale and

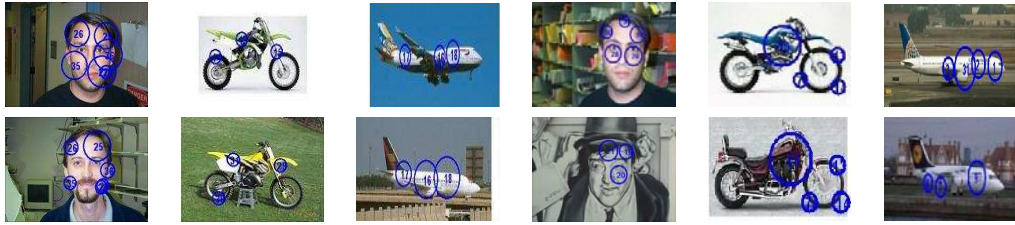

Figure 8: Parsed Results by Hybrid Model (left three panels). Parsed by Standard Model (right three panels).

rotation, (c) that we can learn models for hybrid classes, and (d) that we can perform inference rapidly in under one second.

Our approach can also be extended. By using a richer vocabulary of features we can learn a more sophisticated generative grammar which will be able to represent objects in greater detail and deal with significant variations in viewpoint and appearance.

## Acknowledgements

We gratefully acknowledge support from the W.M. Keck Foundation, NSF grant number 0413214, and NIH grant RO1 EY015261.

## References

[1] D. McAllester, M. Collins, and F. Pereira. Case-Factor Diagrams for Structured Probabilistic Modeling in UAI, 2004.

[2] B.D. Ripley. "Pattern Recognition and Neural Networks". Cambridge University Press. 1996.

[3] J. Lafferty, A. McCallum and F. Pereira. Conditional Random Fields: Probabilistic Models for Segmenting and Labeling Sequence Data. ICML-2001

[4] D. Klein and C. Manning, "Natural Language Grammar Induction Using a Constituent-Context Model," Advances in Neural Information Processing Systems 14 (NIPS-2001), 2001.

[5] H. Dechter and Robert Mateescu. AND/OR Search Spaces for Graphical Models. In Artificial Intelligence, 2006.

[6] H. Chen, Z.J. Xu, Z.Q. Liu, and S.C. Zhu. Composite Templates for Cloth Modeling and Sketching. Proc. IEEE Conf. on Pattern Recognition and Computer Vision, June, New York, 2006.

[7] M. Meila and M. I. Jordan. Mixture of Trees: Learning with mixtures of trees. Journal of Machine Learning Research, 1, 1-48, 2000.

[8] S. Della Pietra, V. Della Pietra, and J. Lafferty. Feature Induction for MRF: Inducing features of random fields. IEEE Transactions on Pattern Analysis and Machine Intelligence, 19(4), April 1997, pp. 380-393

[9] S. C. Zhu, Y. N. Wu, and D. Mumford, "Minimax entropy principle and its application to texture modeling," Neural Comp., vol. 9, no. 8, pp. 1627–1660, 1997.

[10] A. McCallum. Feature Induction for CRF: Efficiently Inducing Features of Conditional Random Fields. Conference on Uncertainty in Artificial Intelligence (UAI), 2003.

[11] L. S. Zettlemoyer and M. Collins. Learning to Map Sentences to Logical Form: Structured Classification with Probabilistic Categorial Grammars. Conference on Uncertainty in Artificial Intelligence (UAI), 2005.

[12] N. Friedman. The Bayesian structural EM algorithm. Fourteenth Conf. on Uncertainty in Artificial Intelligence (UAI), 1998.

[13] T. Kadir and M. Brady. Scale, Saliency and Image Description. International Journal of Computer Vision. 45 (2):83-105, November 2001.

[14] D. G. Lowe, "Distinctive image features from scale-invariant keypoints," International Journal of Computer Vision, 60, 2. pp. 91-110. 2004.

[15] R.J.A. Little and D.B. Rubin. Statistical Analysis with Missing Data, Wiley, Hoboken, New Jersey. 2002.

[16] Fergus, R. , Perona, P. and Zisserman, A. Object Class Recognition by Unsupervised Scale-Invariant Learning Proc. of the IEEE Conf on Computer Vision and Pattern Recognition. 2003.
